# Modeling Time Varying Systems
# Using Hidden Control Neural Architecture

**Esther Levin**

AT&T Bell Laboratories
Speech Research Department
Murray Hill, NJ 07974 USA

## ABSTRACT

Multi-layered neural networks have recently been proposed for non-linear prediction and system modeling. Although proven successful for modeling time invariant nonlinear systems, the inability of neural networks to characterize temporal variability has so far been an obstacle in applying them to complicated nonstationary signals, such as speech. In this paper we present a network architecture, called "Hidden Control Neural Network" (HCNN), for modeling signals generated by nonlinear dynamical systems with restricted time variability. The approach taken here is to allow the mapping that is implemented by a multi layered neural network to change with time as a function of an additional control input signal. This network is trained using an algorithm that is based on "back-propagation" and segmentation algorithms for estimating the unknown control together with the network's parameters. The HCNN approach was applied to several tasks including modeling of time-varying nonlinear systems and speaker-independent recognition of connected digits, yielding a word accuracy of 99.1%.

## I. INTRODUCTION

Layered networks have attracted considerable interest in recent years due to their ability to model adaptively nonlinear multivariate functions. It has been recently proved in [1], that a network with one intermediate layer of sigmoidal units can approximate arbitrarily well any continuous mapping. However, being a static model, a layered network is not capable of modeling signals with an inherent time variability, such as speech.

In this paper we present a hidden control neural network that can implements non-linear and time-varying mapping. The hidden control input signal which allows the network's mapping to change over time, provides the ability to capture the non-stationary properties, and learn the underlying temporal structure of the modeled signal.

## II. THE MODEL

### II.1 MULTI LAYERED NETWORK

Multi layered neural network is a connectionist models that implements a nonlinear mapping from and input $x \in X \subset R^{N_I}$ to an output $y \in Y \subset R^{N_o}$:

$$y = F_\omega(x),\tag{1}$$

where $\omega \in \Omega \subset R^D$, the parameter set of the network, consists of the connection wegihts and the biases, and $x$ and $y$ are the activation vectors of the input and output layers, of dimensionality $N_I$ and $N_O$, respectively.

Recently layered networks have proven useful for non-linear prediction of signals and system modeling [2]. In these applications one uses the values of a real signal $x(t)$, at a set of discrete times in the past, to predict $x(t)$ at a point in the future. For example, for order-one-predictor, the output of the network $y$ is used as a predictor of the next signal sample, when the network is given past sample as input, e.g. $y=\hat{x}_t=F_\omega(x_{t-1})$, where $\hat{x}_t$ denotes the predicted value of the signal at time $t$, which, in general, differs from the true value, $x_t$. The parameter set of the network $\omega$ is estimated from a training set of discrete time samples from a segment of known signal $\{ x_t, t=0, ..., T \}$, by minimizing a prediction error which measures the distortion between the signal and the prediction made by the network,

$$E(\omega)=\sum_{t=1}^{T} \| x_t-F_\omega(x_{t-1}) \|^2,\tag{2}$$

and the estimated parameter set $\hat{\omega}$ is given by $\underset{\Omega}{\operatorname{argmin}} E(\omega)$.

In [2] such a neural network predictor is used for modeling chaotic series. One of the examples considered in [2] is prediction of time series generated by the classic logistic, or Feigenbaum, map,

$$x_{t+1} =4 \cdot b \cdot x_t (1-x_t)\tag{3}$$

This iterated map produces an ergodic chaotic time series when $b$ is chosen to equal 1. Although this time series passes virtually every test for randomness, it is generated by the deterministic Eq.(3), and can be predicted perfectly, once the generating system (3) is learned. Using the back-propagation algorithm [3] to minimize the prediction error (2) defined on a set of samples of this time series, the network parameters $\omega$ were adjusted, enabling accurate prediction of the next point $x_{t+1}$ in this "random" series given the present point $x_t$ as an input. The mapping $F_\omega$ implemented by the trained network approximated very closely the logistic map (3) that generated the modeled series.

### II.2 HIDDEN CONTROL NETWORK

For a given fixed value of the parameters $\omega$, a layered network implements a fixed input-output mapping, and therefore can be used for time-invariant system modeling or prediction of signals generated by a fixed, time-invariant system. Hidden control network that is based on such layered network, has an additional mechanism that allows the mapping (1) to change with time, keeping the parameters $\omega$ fixed. We consider the case where the units in the input layer are divided into two distinct groups. The first input unit group represents the observable input to the network, $x \in X \subset R^p$, and the second represents a control signal $c \in C \subset R^q$, $p+q=N_I$, that controls the mapping between the observable input $x$, and the network output $y$.

The output of the network $y$ is given, according to (1), by $F_\omega(x,c)$, where $(x,c)$ denotes the concatenation of the two inputs. We focus on the mapping between the observable input $x$ and the output. This mapping is modulated by the control input $c$:

for a fixed value of $x$ and for different values of $c$, the network produces different outputs. For a fixed control input, the network implements a fixed observable input-output mapping, but when the control input changes, the network's mapping changes as well, modifying the characteristics of the observed signal:

$$y = F_\omega(x, c) \triangleq F_{\omega, c}(x). \tag{4}$$

If the control signal is known for all time $t$, there is no point in distinguishing between the observable input, $x$, and the control input $c$. The more interesting situation is when the control signal is unknown or hidden, i.e., the *hidden control* case, which we will treat in this paper.

This model can be used for prediction and modeling of nonstationary signals generated by time-varying sources. In the case of first order prediction the present value of the signal $x_t$ is predicted based on $x_{t-1}$, with respect to the control input $c_t$. If we restrict the control signal to take its values from a finite set, $c \in \{C_1, \cdots, C_N\} \equiv C$, then the network is a finite state network, where in each state it implements a fixed input-output mapping $F_{\omega, C_i}$. Such a network with two or more intermidiate layers can approximate arbitrarily closely any set $\{F_1, \cdots, F_N\}$ of continuous functions of the observable input $x$ [4].

In the applications we considered for this model, two types of time-structures were used, namely

**Fully connected model:** In this type of HCNN, every state, corresponding to a specific value of the control input, can be reached from any other state in a single time step. It means that there are no temporal restrictions on the control signal, and in each time step, it can take any of its $N$ possible values $\{C_1, ..., C_N\}$. For example, a 2 state fully connected model is shown in Fig. 1a. In a generative mode of operation, when the observable input of the network is wired to be the the previous network's output , the observable signal $x(t)$ is generated in each one of the states by a different dynamics: $x_{t+1} = F_{c_t}(x_t)$, $c_t \in \{0, 1\}$, and therefore this network emulates two different dynamical systems, with the control signal acting as a switch between them.

**Left-to-right model:** For spoken word modeling, we will consider a finite-state, left-to-right HCNN (see Fig.1b), where the control signal is further restricted to take value $C_i$ only if in the previous time step it had a value of $C_i$ or $C_{i-1}$. Each state of this network represents an unspecified acoustic unit, and due to the "left-to-right" structure, the whole word is modeled as concatenation of such acoustic units. The time spent in each of the states is not fixed, since it varies according to the value of the control signal, and therefore the model can take into account the duration variability between different utterances of the same word.

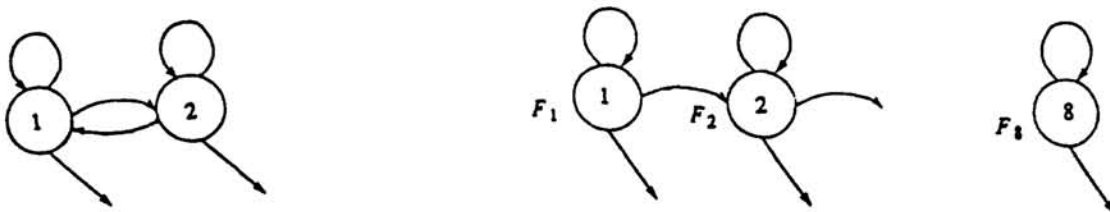

Figure 1: a-Fully connected 2 state HCNN ;
b-Left to right 8 state HCNN for word modeling.

## III. USING HCNN

Given the predictive form of HCNN described in the previous section, there are three basic problems of interest that must be solved for the model to be useful in real-world applications. This problems are the following:

**Segmentation problem** : Here we attempt to uncover the hidden part of the model, i.e., given a network $\omega$ and a sequence of observations { $x_t$, $t=0, ..., T$ }, to find the correct control sequence, which best explains the observations. This problem is solved using an optimality criterion, namely the prediction error, similar to Eq.(2),

$$E(\omega, \mathbf{c}_1^T) = \sum_{t=1}^{T} \| x_t - F_{\omega, c_t}(x_{t-1}) \|^2 ,$$ (5)

where $\mathbf{c}_1^T$ denotes the control sequence $c_1, \cdots, c_T, c_i \in C$. For a given network, $\omega$, the prediction error (5) is a function of the hidden control input sequence, and thus segmentation is associated with the minimization:

$$\hat{\mathbf{c}}_1^T = \underset{\mathbf{c}^T}{argmin} E(\omega, \mathbf{c}_1^T) ,$$ (6)

In the case of a finite-state, fully connected model, this minimization can be performed exhaustively, by minimizing for each observation separately, and for a fully connected HCNN with a real-valued control signal (i.e. not the finite state case), local minimization of (5) can be performed using the back-propagation algorithm. For a "left-to-right" model , global minimum of (5) is attained efficiently using the Viterbi algorithm [5].

**Evaluation problem,** namely how well a given network $\omega$ matches a given sequence of observations { $x_t$, $t=0, ..., T$ }. The evaluation is a key point for many applications. For example, if we consider the case in which we are trying to choose among several competing networks, that represent different hypothesis in the hypotheses space, the solution to Problem 2 allows us to choose the network that best matches the observation. This problem is also solved using the prediction error defined in (5). The match, or actually, the distortion, is measured by the prediction error of the network on a sequence of observations, for the best possible sequence of hidden control inputs, i.e.,

$$E(\omega) = \underset{\mathbf{c}^T}{min} E(\omega, \mathbf{c}_1^T) .$$ (7)

Therefore, to evaluate a network, first the segmentation problem must be solved.

**Training problem,** i.e., how to adjust the model parameters $\omega$ to best match the observation sequence, or training set, { $x_t$, $t=0, ..., T$ }.

The training in layered networks is accomplished by minimizing the prediction error of Eq.(2) using versions of the back-propagation algorithm. In the HCNN case, the prediction error (5) is a function of the hidden parameters and the hidden control input sequence, and thus training is associated with the joint minimization:

$$\hat{\omega} = \underset{\Omega}{argmin} \{ \underset{\mathbf{c}^T}{min} E(\omega, \mathbf{c}_1^T) \} .$$ (8)

This minimization is performed by an iterative training algorithm.

The $k$-th iteration of the algorithm consists of two stages:

**1. Reestimation:** For the present value of the control input sequence, the prediction error is minimized with respect to the network parameters.

$$(\hat{\omega})_k = \underset{\Omega}{argmin} E(\omega, (\mathbf{c}_1^T)_{k-1})$$ (9)

This minimization is implemented by the back-propagation algorithm.

**2. Segmentation:** Using the values of parameters, obtained from the previous stage, the control sequence is estimated (as in (6) ).

$$(c_1^T)_k = \underset{c^T}{argmin} \, E((\hat{\omega})_k, c_1^T) \qquad (10)$$

# IV. HCNN AS A STATISTICAL MODEL

For further understanding of the properties of the proposed model and the training procedure, it is useful to describe the HCNN by an equivalent statistical vector source of the following form:

$$x_t = F_{\omega, c_t}(x_{t-1}) + n_t, \quad n_t \sim N(0, I), \qquad (11)$$

where $n_t$ is a white Gaussian noise. Assuming for simplicity that all the values of the control allowed by the model are equiprobable (this is a special case of Markov process, and can be easily extended for the general case) , we can write the joint likelihood of the data and the control

$$P(x_1^T, c_1^T \mid \omega) = (2\pi)^{-\frac{pT}{2}} \exp[-\frac{1}{2}\sum_{t=1}^{T} \| x_t - F_{\omega, c_t}(x_{t-1}) \|^2], \qquad (12)$$

where $x_1^T$ denotes the sequence of observation $\{x_1, x_2, \cdots, x_T\}$.

Eq.(12) provides a probabilistic interpretation of the procedures described in the previous section:

The proposed segmentation procedure is equivalent to choosing the most probable control sequence, given the network and the observations.

The evaluation of the network is related to the probability of the observations given the model, for the best sequence of control inputs,

$$\underset{c^T}{min} \, E(\omega, c_1^T) <=> \underset{c^T}{max} \, P(x_1^T, c_1^T \mid \omega), \qquad (13)$$

The proposed training procedure (Eq. 8) is equivalent to maximization of the joint likelihood (12):

$$\hat{\omega} = argmin\{ \underset{c^T}{min} \, E(\omega, s_1^T) \} = argmax\{ \underset{c^T}{max} \, P(x_1^T, c_1^T \mid \omega) \}. \qquad (14)$$

Thus (8) is equivalent to an approximate maximum likelihood training, where instead of maximizing the marginal likelihood $P(x_1^T \mid \omega) = \sum_{c^T} P(x_1^T, c_1^T \mid \omega)$, only the maximal term in the sum, the joint likelihood (14) is considered. The approximate maximum likelihood training avoids the computational complexity of the exact maximum likelihood approach, and recently [6] was shown to yield results similar to those obtained by the exact maximum likelihood training.

## IV.1 HCNN and the Hidden Markov Model (HMM)

During the past decade hidden Markov modeling has been used extensively to represent the probability distribution of spoken words [7]. A hidden Markov model assumes that the modeled speech signal can be characterized as being produced at each time instant by one of the states of a finite state source, and that each observation vector is an independent sample according to the probability distribution of the current state. The transitions between the states of the model are governed by a Markov process

HCNN can be viewed as an extension of this model to the case of Markov output processes. The observable signal in each state is modeled as though it was produced by

a dynamical system driven by noise. Here we are modeling the dynamics that generated the signal, $F_\omega$, and the dependence of the present observation vector on the previous one. The assumption that the driving noise (12) is normal is not necessary: instead, we can assume a parametric form of the noise density, and estimate its parameters.

## V. EXPERIMENTAL EVALUATION

For experimental evaluation of the proposed model, we tested on two different tasks:

### V.1 Time-varying system modeling and segmentation

Here an HCNN was used for a single-step prediction of a signal generated by a time-varying system, described by

$$x_{t+1} = \begin{cases} F_L(x_t) & \text{if } switch = 0 \\ 1 - F_L(x_t) & \text{if } switch = 1 \end{cases}, \tag{15}$$

where $F_L$ is the logistic map from Eq. (3), and *switch* is a random variable, assuming binary values. Both of the systems, $F_L$, and $1-F_L$, are chaotic and produce signals in the range [0,1]. A fully connected, 2-state HCNN (each state corresponding to one switch position), as in Fig. 1a, was trained on a segment of 400 samples of such a signal, according to the training algorithm described in section V. The performance of the resulting network was tested on an independent set of 1000 samples of this signal. The estimated control sequence differed from the real switch position in only 8 out of 1000 test samples. The evaluation score, i.e., the average prediction error for this estimated control sequence was $7.5\times10^{-5}$ per sample. Fig. 2 compares the mapping implemented by the network in one state, corresponding to control value set to 0, and the logistic map for *switch*=0. Similar results are obtained for $c=1$ and *switch*=1. These results indicate that the HCNN was indeed able to capture the two underlying dynamics that generated the modeled signal, and to learn the switching pattern simultaneously.

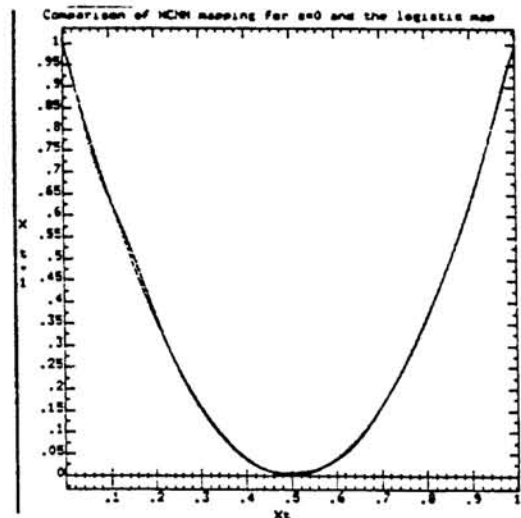

Fig.2 Comparison of the logistic map and the mapping implemented by HCNN with $c=0$.

### V.2 Continuous recognition of digit sequences

Here we tested the proposed HCNN modeling technique on recognition of connected spoken versions of the digits, consisting of "zero" to "nine", and including the word "oh", recorded from male speakers through a telephone handset and sampled at 6.67

kHz. LPC analysis of order 8 was performed on frames of 45 msec duration, with overlap of 15 msec, and 12 cepstral and 12 delta cepstral [8] coefficients were derived for the $t$-th frame to form the observable signal $x_t$. Each digit was modeled by an 8 state, left-to-right HCNN, as in Fig.1b. The network was trained to predict the cepstral and delta cepstral coefficients for the next frame. Each network consisted of 32 input units (24 to encode $x_t$ and 8 for a distributed representation of the 8 control values), 24 output units and 30 hidden units, all fully connected. Each network was trained using a training set of 900 utterances from 44 male speakers extracted from continuous strings of digits using an HMM based recognizer [9]. 1666 strings (5600 words), uttered by an independent set of 22 male speakers were used for estimating the recognition accuracy. The mean and the covariance of the driving noise (12) were modeled. The word accuracy obtained was 99.1%.

Fig. 3a illustrates the process of recognition (the forward pass of Viterbi algorithm) of the word "one" by the speaker-independent system. The horizontal axis is time (in frames). 11 models from "zero" to "nine", and "oh" appear on the vertical axis. The numbers that appear in the graph (from 1 to 8) describe the number of a state. For example, number 2 inside the second row of the graph denotes state number 2 of the model of the word "one". In each frame, the prediction error was calculated for each one of the states in each model, resulting in 88 different prediction errors. The graph in each frame shows the states of the models that are in the vicinity of the minimal error among those 88. This is a partial description of a forward pass of the Viterbi algorithm in recognition, before the left-to-right constraints of the models are taken into account. Figure 3a shows that the main candidate considered in recognition of the word "one" is the actual model of "one", but in the end of the word two spurious candidates arise. The spurious candidates are certain states of the models of "seven" and "nine". Those states are detectors of the nasal 'n' that appears in all these words.

Figure 3b shows the recognition of a four digit string "three - five - oh - four". The spurious candidates indicate detectors of certain sounds, common to different words, like in "four" and in "oh", in "five" and in "nine", in "three", "six" and "eight".

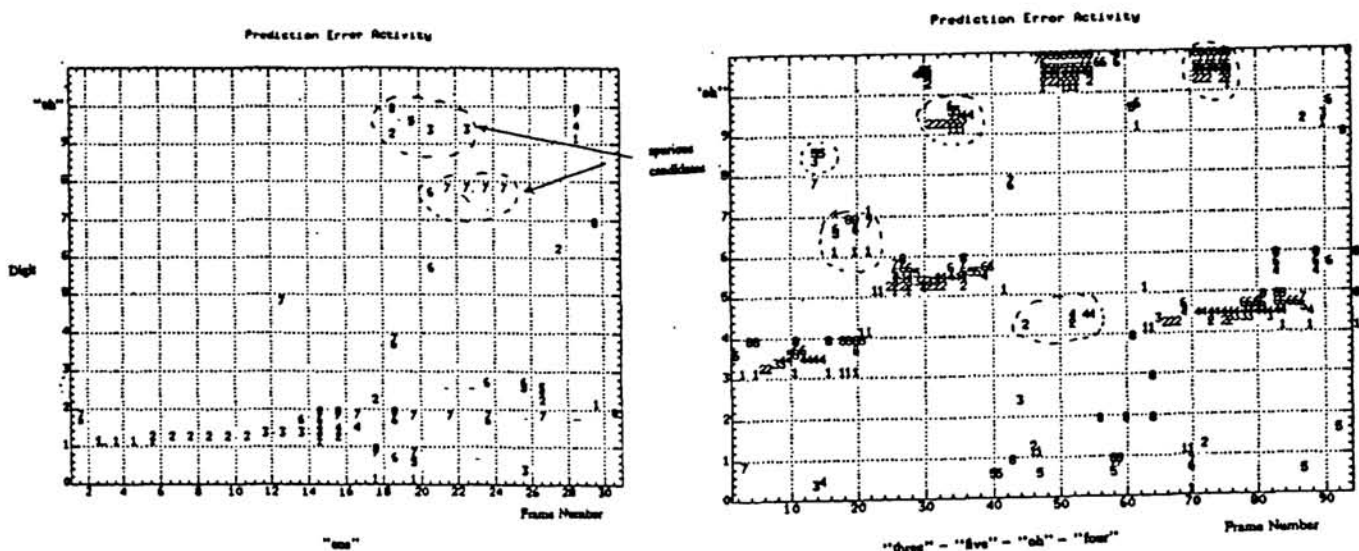

Fig. 3 Illustration of the recognition process.

## VI. SUMMARY AND DISCUSSION

This paper introduces a generalization of the layered neural network that can implement a time-varying non-linear mapping between its observable input and output. The variation of the network's mapping is due to an additional, hidden control input, while the network parameters remain unchanged. We proposed an algorithm for finding the network parameters and the hidden control sequence from a training set of examples of observable input and output. This algorithm implements an approximate maximum likelihood estimation of parameters of an equivalent statistical model, when only the dominant control sequence is taken into account. The conceptual difference between the proposed model and the HMM is that in the HMM approach, the observable data in each of the states is modeled as though it was produced by a memoryless source, and a parametric description of this source is obtained during training, while in the proposed model the observations in each state are produced by a non-linear dynamical system driven by noise, and both the parametric form of the dynamics and the noise are estimated. The performance of the model was illustrated for the tasks of nonlinear time-varying system modeling and continuously spoken digit recognition. The reported results show the potential of this model for providing high performance speech recognition capability.

### Acknowledgment

Special thanks are due to N. Merhav for numerous comments and helpful discussions. Useful discussions with N.Z. Tishby, S.A. Solla, L.R. Rabiner and J.G. Wilpon are greatly appreciated.

### References
1. G. Cybenko, " Approximation by superposition of sigmoidal function," *Math. Control Systems Signals,* in press, 1989.
2. A. Lapedes and R. Farber, " Nonlinear signal processing using neural networks: prediction and system modeling, " *Proc of IEEE,* in press, 1989.
3. D.E. Rumelhart, G.E. Hinton and R.J. Williams, "Learning internal representation by error propagation," *Parallel Distributed Processing: Exploration in the Microstructure of Cognition,* MIT Press, 1986.
4. E. Levin, "Word recognition using hidden control neural architecture," *Proc. of ICASSP,* Albuquerque, April 1990.
5. G.D. Forney, "The Viterbi algorithm," *Proc. IEEE,* vol. 61, pp. 268-278, Mar. 1973.
6. N. Merhav and Y. Ephraim, "Maximum likelihood hidden Markov modeling using a dominant sequence of states," accepted for publication in *IEEE Transaction on ASSP.*
7. L. R. Rabiner, "A tutorial on hidden Markov models and selected applications in speech recognition," *Proc. of IEEE,* vol. 77, No. 2, pp. 257-286, February 1989
8. B.S. Atal, "Effectiveness of linear prediction characteristics of the speech wave for automatic speaker identification and verification," *J. Acoust. Soc. Am.,* vol. 55, No. 6, pp. 1304-1312, June 1974.
9. L.R. Rabiner, J.G. Wilpon, and F.K. Soong, "High performance connected digit recognition using hidden Markov models," *IEEE Transaction on ASSP,* vol. 37, 1989.